# Directed Regression

**Yi-hao Kao**
Stanford University
Stanford, CA 94305
yihaokao@stanford.edu

**Benjamin Van Roy**
Stanford University
Stanford, CA 94305
bvr@stanford.edu

**Xiang Yan**
Stanford University
Stanford, CA 94305
xyan@stanford.edu

## Abstract

When used to guide decisions, linear regression analysis typically involves estimation of regression coefficients via ordinary least squares and their subsequent use to make decisions. When there are multiple response variables and features do not perfectly capture their relationships, it is beneficial to account for the decision objective when computing regression coefficients. Empirical optimization does so but sacrifices performance when features are well-chosen or training data are insufficient. We propose *directed regression*, an efficient algorithm that combines merits of ordinary least squares and empirical optimization. We demonstrate through a computational study that directed regression can generate significant performance gains over either alternative. We also develop a theory that motivates the algorithm.

## 1  Introduction

When used to guide decision-making, linear regression analysis typically treats estimation of regression coefficients separately from their use to make decisions. In particular, estimation is carried out via ordinary least squares (OLS) without consideration of the decision objective. The regression coefficients are then used to optimize decisions.

When there are multiple response variables and features do not perfectly capture their relationships, it is beneficial to account for the decision objective when computing regression coefficients. Imperfections in feature selection are common since it is difficult to identify the right features and the number of features is typically restricted in order to avoid over-fitting.

Empirical optimization (EO) is an alternative to OLS which selects coefficients that minimize empirical loss in the training data. Though it accounts for the decision objective when computing regression coefficients, EO sacrifices performance when features are well-chosen or training data is insufficient.

In this paper, we propose a new algorithm – directed regression (DR) – which is a hybrid between OLS and EO. DR selects coefficients that are a convex combination of those that would be selected by OLS and those by EO. The weights of OLS and EO coefficients are optimized via cross-validation.

We study DR for the case of decision problems with quadratic objective functions. The algorithm takes as input a training set of data pairs, each consisting of feature vectors and response variables, together with a quadratic loss function that depends on decision variables and response variables. Regression coefficients are computed for subsequent use in decision-making. Each future decision depends on newly sampled feature vectors and is made prior to observing response variables with the goal of minimizing expected loss.

We present computational results demonstrating that DR can substantially outperform both OLS and EO. These results are for synthetic problems with regression models that include subsets of relevant

features. In some cases, OLS and EO deliver comparable performance while DR reduces expected loss by about $20\%$. In none of the cases considered does either OLS or EO outperform DR.

We also develop a theory that motivates DR. This theory is based on a model in which selected features do not perfectly capture relationships among response variables. We prove that, for this model, the optimal vector of coefficients is a convex combination of those that would be generated by OLS and EO.

## 2  Linear Regression for Decision-Making

Suppose we are given a set of training data pairs $O = \{(x^{(1)}, y^{(1)}), \cdots, (x^{(N)}, y^{(N)})\}$. Each $n$th data pair is comprised of feature vectors $x_1^{(n)}, \ldots, x_K^{(n)} \in \Re^M$ and a vector $y^{(n)} \in \Re^M$ of response variables. We would like to compute regression coefficients $r \in \Re^K$ so that given a data pair $(x, y)$, the linear combination $\sum_k r_k x_k$ of feature vectors estimates the expectation of $y$ conditioned on $x$. We restrict attention to cases where $M > 1$, with special interest in problems where $M$ is large, because it is in such situations that DR offers the largest performance gains.

We consider a setting where the regression model is used to guide future decisions. In particular, after computing regression coefficients, each time we observe feature vectors $x_1, \ldots, x_K$ we will have to select a decision $u \in \Re^L$ before observing the response vector $y$. The choice incurs a loss

$$\ell(u, y) = u^\top G_1 u + u^\top G_2 y,$$

where the matrices $G_1 \in \Re^{L \times L}$ and $G_2 \in \Re^{L \times M}$ are known, and the former is positive definite and symmetric. We aim to minimize expected loss, assuming that the conditional expectation of $y$ given $x$ is $\sum_{k=1}^K r_k x_k$. As such, given $x$ and $r$, we select a decision

$$u_r(x) = \operatorname*{argmin}_u \ell\left(u, \sum_{k=1}^K r_k x_k\right) = -\frac{1}{2} G_1^{-1} G_2 \sum_{k=1}^K r_k x_k.$$

The question is how best to compute the regression coefficients $r$ for this purpose.

To motivate the setting we have described, we offer a hypothetical application.

**Example 1.** *Consider an Internet banner ad campaign that targets $M$ classes of customers. An average revenue of $y_m$ is received per customer of class $m$ that the campaign reaches. This quantity is random and influenced by $K$ observable factors $x_{1m}, \ldots, x_{Km}$. These factors may be correlated across customers classes; for example, they could capture customer preferences as they relate to ad content or how current economic conditions affect customers. For each $m$th class, the cost of reaching the $u_m$th customer increases with $u_m$ because ads are first targeted at customers that can be reached at lower cost. This cost is quadratic, so that we pay $\gamma_m u_m^2$ to reach $u_m$ customers, where $\gamma_m$ is a known constant.*

*The application we have described fits our general problem context. It is natural to predict the response vector $y$ using a linear combination $\sum_k r_k x_k$ of factors with the regression coefficients $r_k$ computed based on past observations $O = \{(x^{(1)}, y^{(1)}), \cdots, (x^{(N)}, y^{(N)})\}$. The goal is to maximize expected revenue less advertising costs. This gives rise to a loss function that is quadratic in $u$ and $y$:*

$$\ell(u, y) = \sum_{m=1}^M (\gamma_m u_m^2 - u_m y_m).$$

One might ask why not construct $M$ separate linear regression models, one for each response variable, each with a separate set of $K$ coefficients. The reason is that this gives rise to $MK$ coefficients; when $M$ is large and data is limited, this could lead to over-fitting. Models of the sort we consider, where regression coefficients are shared across multiple response variables, are sometimes referred to as *general linear models* and have seen a wide range of applications [7, 8]. It is well-known that the quality of results is highly sensitive to the choice of features, even more so than for models involving a single response variable [7].

## 3 Algorithms

Ordinary least squares (OLS) is a conventional approach to computing regression coefficients. This would produce a coefficient vector

$$r^{\text{OLS}} = \underset{r \in \Re^K}{\operatorname{argmin}} \sum_{n=1}^{N} \left\| y^{(n)} - \sum_{k=1}^{K} r_k x_k^{(n)} \right\|^2. \tag{1}$$

Note that OLS does not take the decision objective into account when computing regression coefficients. Empirical optimization (EO), as studied for example in [2, 6], offers an alternative that does so. This approach minimizes empirical loss on the training data:

$$r^{\text{EO}} = \underset{r \in \Re^K}{\operatorname{argmin}} \sum_{n=1}^{N} \ell(u_r(x^{(n)}), y^{(n)}). \tag{2}$$

Note that EO does not explicitly aim to estimate the conditional expectation of the response vector. Instead it focusses on decision loss that would be incurred with the training data. Both $r^{\text{OLS}}$ and $r^{\text{EO}}$ can be computed efficiently by minimizing convex quadratic functions.

As we will see in our computational and theoretical analyses, OLS and EO can be viewed as two extremes, each offering room for improvement. In this paper, we propose an alternative algorithm – directed regression (DR) – which produces a convex combination $r^{\text{DR}} = (1 - \lambda) r^{\text{OLS}} + \lambda r^{\text{EO}}$ of coefficients computed by OLS and EO. The term *directed* is chosen to indicate that DR is influenced by the decision objective though, unlike EO, it does not simply minimize empirical loss. The parameter $\lambda \in [0, 1]$ is computed via cross-validation, with an objective of minimizing average loss on validation data. Average loss is a convex quadratic function of $\lambda$, and therefore can be easily minimized over $\lambda \in [0, 1]$.

DR is designed to generate decisions that are more robust to imperfections in feature selection than OLS. As such, DR addresses issues similar to those that have motivated work in data-driven robust optimization, as surveyed in [3]. Our focus on making good decisions despite modeling inaccuracies also complements recent work that studies how models deployed in practice can generate effective decisions despite their failure to pass basic statistical tests [4].

## 4 Computational Results

In this section, we present results from applying OLS, EO, and DR to synthetic data. To generate a data set, we first sample parameters of a generative model as follows:

1. Sample $P$ matrices $C_1, \ldots, C_P \in \Re^{M \times Q}$, with each entry from each matrix drawn independently from $\mathcal{N}(0, 1)$.

2. Sample a vector $\tilde{r} \in \Re^P$ from $\mathcal{N}(0, I)$.

3. Sample $G_a \in \Re^{L \times L}$ and $G_b \in \Re^{L \times M}$, with each entry of each matrix drawn from $\mathcal{N}(0, 1)$. Let $G_1 = G_a^\top G_a$ and $G_2 = G_a^\top G_b$.

Given generative model parameters $C_1, \ldots, C_P$ and $\tilde{r}$, we sample each training data pair $(x^{(n)}, y^{(n)})$ as follows:

1. Sample a vector $\phi^{(n)} \in \Re^Q$ from $\mathcal{N}(0, I)$ and a vector $w^{(n)} \in \Re^M$ from $\mathcal{N}(0, \sigma_w^2 I)$.

2. Let $y^{(n)} = \sum_{i=1}^{P} \tilde{r}_i C_i \phi^{(n)} + w^{(n)}$.

3. For each $k = 1, 2, \cdots, K$, let $x_k^{(n)} = C_k \phi^{(n)}$.

The vector $\phi^{(n)}$ can be viewed as a sample from an underlying information space. The matrices $C_1, \ldots, C_P$ extract feature vectors from $\phi^{(n)}$. Note that, though response variables depend on $P$ feature vectors, only $K \leq P$ are used in the regression model.

Given generative model parameters and a coefficient vector $r \in \Re^K$, it is easy to evaluate the expected loss $\bar{\ell}(r) = \mathrm{E}_{x,y}[\ell(u_r(x), y)]$. It is also easy to evaluate the minimal expected loss $\ell^* =$

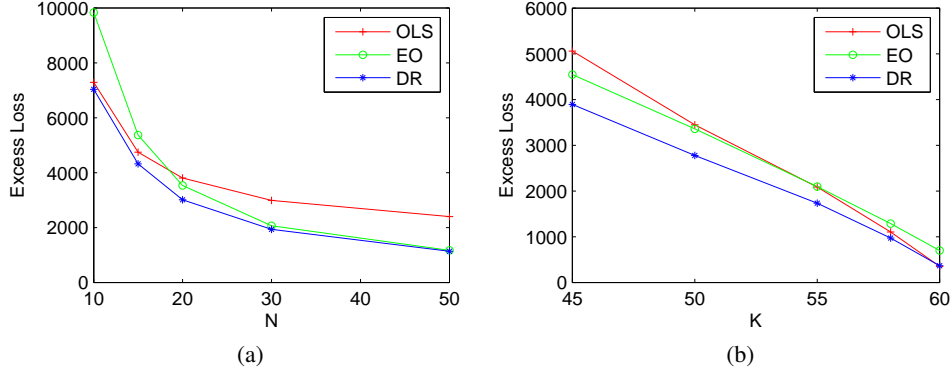

Figure 1: (a) Excess losses delivered by OLS, EO, and DR, for different numbers $N$ of training samples. (b) Excess losses delivered by OLS, EO, and DR, using different numbers $K$ of the 60 features.

$\min_r \mathrm{E}_{x,y}[\ell(u_r(x), y)]$. We will assess each algorithm in terms of the excess loss $\bar{\ell}(r) - \ell^*$ delivered by the coefficient vector $r$ that the algorithm computes. Excess loss is nonnegative, and this allows us to make comparisons in percentage terms.

We carried out two sets of experiments to compare the performance of OLS, EO, and DR. In the first set, we let $M = 15$, $L = 15$, $P = 60$, $Q = 20$, $\sigma_w = 5$, and $K = 50$. For each $N \in \{10, 15, 20, 30, 50\}$, we ran 100 trials, each with an independently sampled generative model and training data set. In each trial, each algorithm computes a coefficient vector given the training data and loss function. With DR, $\lambda$ is selected via leave-one-out cross-validation when $N \leq 20$, and via 5-fold cross-validation when $N > 20$. Figure 1(a) plots excess losses averaged over trials. Note that the excess loss incurred by DR is never larger than that of OLS or EO. Further, when $N = 20$, the excess loss of OLS and EO are both around 20% larger than that of DR. For small $N$, OLS is as effective as DR, while, EO becomes as effective as DR as $N$ grows large.

In the second set of experiments, we use the same parameter values as in the first set, except we fix $N = 20$ and consider use of $K \in \{45, 50, 55, 58, 60\}$ feature vectors. Again, we ran 100 trials for each $K$, applying the three algorithms as in the first set of experiments. Figure 1(b) plots excess losses averaged over trials. Note that when $K = 55$, DR delivers excess loss around 20% less than EO and OLS. When $K = P = 60$, there are no missing features and OLS matches the performance of DR.

Figure 2 plots the values of $\lambda$ selected by cross-validation, each averaged over the 100 trials, as a function of $N$ and $K$. As the number of training samples $N$ grows, so does $\lambda$, indicating that DR is weighted more heavily toward EO. As the number of feature vectors $K$ grows, $\lambda$ diminishes, indicating that DR is weighted more heavily toward OLS.

## 5 Theoretical Analysis

In this section, we formulate a generative model for the training data and future observations. For this model, optimal coefficients are convex combinations of $r^{\mathrm{OLS}}$ and $r^{\mathrm{EO}}$. As such, our model and analysis motivate the use of DR.

### 5.1 Model

In this section, we describe a generative model that samples the training data set, as well as "missing features," and a representative future observation. We then formulate an optimization problem where the objective is to minimize expected loss on the future observation conditioned on the training data and missing features. It may seem strange to condition on missing features since in practice they are unavailable when computing regression coefficients. However, we will later establish that optimal

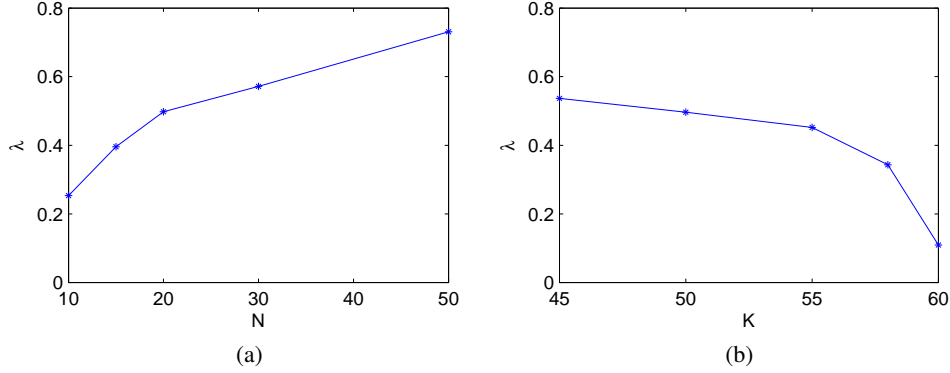

Figure 2: (a) The average values of selected $\lambda$, for different numbers $N$ of training samples. (b) The average values of selected $\lambda$, using different numbers $K$ of the 60 features.

coefficients are convex combinations of $r^{\mathrm{OLS}}$ and $r^{\mathrm{EO}}$, each of which can be computed without observing missing features. Since directed regression searches over these convex combinations, it should approximate what would be generated by a hypothetical algorithm that observes missing features.

We will assume that each feature, whether observed or missing, is a linear function of an "information vector" drawn from $\Re^Q$. Specifically, the $N$ training data samples depend on information vectors $\phi^{(1)}, \ldots, \phi^{(N)} \in \Re^Q$. A linear function mapping an information vector to a feature vector can be represented by a matrix in $\Re^{M \times Q}$, and to describe our generative model, it is useful to define an inner product for such matrices. In particular, we define the inner product between matrices $A$ and $B$ by

$$\langle A, B \rangle = \frac{1}{N} \sum_{n=1}^{N} (A\phi^{(n)})^\top (B\phi^{(n)}).$$

Our generative model takes several parameters as input. First, there are the number of samples $N$, the number of response variables $M$, and the number of feature vectors $K$. Second, a parameter $\mu_Q$ specifies the expected dimension of the information vector. Finally, there are standard deviations $\sigma_r, \sigma_\epsilon$, and $\sigma_w$, of observed feature coefficients, missing feature coefficients, and noise, respectively. Given parameters $N$, $M$, $K$, $\mu_Q$, $\sigma_r$, $\sigma_\epsilon$, and $\sigma_w$, the generative model produces data as follows:

1. Sample $Q$ from the geometric distribution with mean $\mu_Q$.

2. Sample $\phi^{(1)}, \ldots, \phi^{(N)} \in \Re^Q$ from $\mathcal{N}(0, I_Q)$.

3. Sample $C_1, \ldots, C_K$ and $D_1, \cdots, D_J \in \Re^{M \times Q}$ with each entry i.i.d. from $\mathcal{N}(0,1)$, where $K + J = MQ$.

4. Apply the Gram-Schmidt algorithm with respect to the inner product defined above to generate an orthonormal basis $\tilde{C}_1, \ldots, \tilde{C}_K, \tilde{D}_1, \ldots, \tilde{D}_J$ from the sequence $C_1, \ldots, C_K, D_1, \ldots, D_J$.

5. Sample $r^* \in \Re^K$ from $\mathcal{N}(0, \sigma_r^2 I_K)$ and $r^{\perp *} \in \Re^J$ from $\mathcal{N}(0, \sigma_\epsilon^2 I_J)$.

6. For $n = 1, \cdots, N$, sample $w^{(n)} \in \Re^M$ from $\mathcal{N}(0, \sigma_w^2 I_M)$, and let

$$x^{(n)} = \begin{bmatrix} C_1 \phi^{(n)} & \cdots & C_K \phi^{(n)} \end{bmatrix}, \tag{3}$$

$$z^{(n)} = \begin{bmatrix} \tilde{D}_1 \phi^{(n)} & \cdots & \tilde{D}_J \phi^{(n)} \end{bmatrix}, \tag{4}$$

$$y^{(n)} = \sum_{k=1}^{K} r_k^* x_k^{(n)} + \sum_{j=1}^{J} r_j^{\perp *} z_j^{(n)} + w^{(n)}. \tag{5}$$

7. Sample $\tilde{\phi}$ uniformly from $\{\phi^{(1)}, \cdots, \phi^{(N)}\}$ and $\tilde{w} \in \Re^M$ from $\mathcal{N}(0, \sigma_w^2 I_M)$. Generate $\tilde{x}$, $\tilde{z}$, and $\tilde{y}$ by the same functions in (3), (4), and (5).

The samples $z^{(1)}, \ldots, z^{(N)}, \tilde{z}$ represent missing features. The Gram-Schmidt procedure ensures two properties. First, since $\langle C_k, \tilde{D}_j \rangle = 0$, missing features are uncorrelated with observed features. If this were not the case, observed features would provide information about missing features. Second, since $\tilde{D}_1, \ldots, \tilde{D}_J$ are orthonormal, the distribution of missing features is invariant to rotations in the $J$-dimensional subspace from which they are drawn. In other words, all directions in that space are equally likely.

We define an augmented training set $\overline{O} = \{(x^{(1)}, z^{(1)}, y^{(1)}), \cdots, (x^{(N)}, z^{(N)}, y^{(N)})\}$ and consider selecting regression coefficients $\hat{r} \in \Re^K$ that solve

$$\min_{r \in \Re^K} \mathrm{E}[\ell(u_r(\tilde{x}), \tilde{y}) | \overline{O}].$$

Note that the probability distribution here is implicitly defined by our generative model, and as such, $\hat{r}$ may depend on $N$, $M$, $K$, $\mu_Q$, $\sigma_r$, $\sigma_\epsilon$, $\sigma_w$, and $\overline{O}$.

## 5.2 Optimal Solutions

Our primary interest is in cases where prior knowledge about the coefficients $r^*$ is weak and does not significantly influence $\hat{r}$. As such, we will from here on restrict attention to the case where $\sigma_r$ is asymptotically large. Hence, $\hat{r}$ will no longer depend on $\sigma_r$.

It is helpful to consider two special cases. One is where $\sigma_\epsilon = 0$ and the other is where $\sigma_\epsilon$ is asymptotically large. We will refer to $\hat{r}$ in these extreme cases as $\hat{r}_0$ and $\hat{r}_\infty$. The following theorem establishes that these extremes are delivered by OLS and EO.

**Theorem 1.** *For all $N$, $M$, $K$, $\mu_Q$, $\sigma_w$, and $\overline{O}$,*

$$\hat{r}_0 = \underset{r \in \Re^K}{\mathrm{argmin}} \sum_{n=1}^{N} \left\| y^{(n)} - \sum_{k=1}^{K} r_k x_k^{(n)} \right\|^2$$

*and*

$$\hat{r}_\infty = \underset{r \in \Re^K}{\mathrm{argmin}} \sum_{n=1}^{N} \ell(u_r(x^{(n)}), y^{(n)}).$$

Note that $\sigma_\epsilon$ represents the degree of bias in a regression model that assumes there are no missing features. Hence, the above theorem indicates that OLS is optimal when there is no bias while EO is optimal as the bias becomes asymptotically large. It is also worth noting that the coefficient vectors $\hat{r}_0$ and $\hat{r}_\infty$ can be computed without observing the missing features, though $\hat{r}$ is defined by an expectation that is conditioned on their realizations. Further, computation of $\hat{r}_0$ and $\hat{r}_\infty$ does not require knowledge of $Q$ or $\sigma_w$.

Our next theorem establishes that the coefficient vector $\hat{r}$ is always a convex combination of $\hat{r}_0$ and $\hat{r}_\infty$.

**Theorem 2.** *For all $N$, $M$, $K$, $\mu_Q$, $\sigma_w$, $\sigma_\epsilon$, and $\overline{O}$,*

$$\hat{r} = (1 - \lambda)\hat{r}_0 + \lambda \hat{r}_\infty,$$

*where $\lambda = \dfrac{1}{1 + \frac{\sigma_w^2}{N\sigma_\epsilon^2}}$.*

Our two theorems together imply that, with an appropriately selected $\lambda \in [0, 1]$, $(1 - \lambda)r^{OLS} + \lambda r^{EO} = \hat{r}$. This suggests that directed regression, which optimizes $\lambda$ via cross-validation to generate a coefficient vector $r^{DR} = (1 - \lambda)r^{OLS} + \lambda r^{EO}$, should approximate $\hat{r}$ well without observing the missing features or requiring knowledge of $Q$, $\sigma_\epsilon$, or $\sigma_w$.

## 5.3 Interpretation

To develop intuition for our results, we consider an idealized situation where the coefficients $r^*$ and $r^{\perp*}$ are provided to us by an oracle. Then the optimal coefficient vector would be

$$r^O = \underset{r \in \Re^K}{\operatorname{argmin}} \operatorname{E}[\ell(u_r(\tilde{x}), \tilde{y}) | \overline{O}, r^*, r^{\perp*}].$$

It can be shown that $r^{\mathrm{OLS}}$ is a biased estimator of $r^O$, while $r^{\mathrm{EO}}$ is an unbiased one. However, the variance of $r^{\mathrm{OLS}}$ is smaller than that of $r^{\mathrm{EO}}$. The optimal tradeoff is indeed captured by the value of $\lambda$ provided in Theorem 2. In particular, as the number of training samples $N$ increases, variance diminishes and $\lambda$ approaches 1, placing increasing weight on EO. On the other hand, as the number of observed features $K$ increases, model bias decreases and $\lambda$ approaches 0, placing increasing weight on OLS. Our experimental results demonstrate that the value of $\lambda$ selected by cross-validation exhibits the same behavior.

## 6 Extensions

Though we only treated linear models and quadratic objective functions, our work suggests that there can be significant gains in broader problem settings from a tighter coupling between machine learning and decision-making. In particular, machine learning algorithms should factor decision objectives into the learning process. It will be interesting to explore how to do this with other classes of models and objectives.

One might argue that feature mis-specification is not a critical issue in light of effective methods for subset selection. In particular, rather than selecting a few features and facing the consequences of model bias, one might select an enormous set of features and apply a method like the lasso [10] to identify a small subset. Our view is that even this enormous set will result in model biases that might be ameliorated by generalizations of DR. There is also the concern that data requirements grow with the size of the large feature set, albeit slowly. Understanding how to synthesize DR with subset selection methods is an interesting direction for future research.

Another issue that should be explored is the effectiveness of cross-validation in optimizing $\lambda$. In particular, it would be helpful to understand how the estimate relates to the ideal value of $\lambda$ identified by Theorem 2. More general work on the selection of convex combinations of models (e.g., [1, 5]) may lend insights to our setting.

Let us close by mentioning that the ideas behind DR ought to play a role in reinforcement learning (RL) as presented in [9]. RL algorithms learn from experience to predict a sum of future rewards as a function of a state, typically by fitting a linear combination of features of the state. This so-called approximate value function is then used to guide sequential decision-making. The problem we addressed in this paper can be viewed as a single-period version of RL, in the sense that each decision incurs an immediate cost but bears no further consequences. It would be interesting to extend our idea to the multi-period case.

## Acknowledgments

We thank James Robins for helpful comments and suggestions. The first author is supported by a Stanford Graduate Fellowship. This research was supported in part by the National Science Foundation through grant CMMI-0653876.

## Appendix

*Proof of Theorem 1.* For each $n$, let $x^{(n)} = \begin{bmatrix} x_1^{(n)} & \cdots & x_K^{(n)} \end{bmatrix}$, $z^{(n)} = \begin{bmatrix} z_1^{(n)} & \cdots & z_J^{(n)} \end{bmatrix}$. Let $X = \begin{bmatrix} x^{(1)\top} & \cdots & x^{(N)\top} \end{bmatrix}^\top$, $Z = \begin{bmatrix} z^{(1)\top} & \cdots & z^{(N)\top} \end{bmatrix}^\top$, $Y = \begin{bmatrix} y^{(1)\top} & \cdots & y^{(N)\top} \end{bmatrix}^\top$, $\bar{r} = \operatorname{E}[r^*|\overline{O}]$, $\bar{r}^\perp = \operatorname{E}[r^{\perp*}|\overline{O}]$. For any matrix $V$, let $V^\dagger$ denote $(V^\top V)^{-1} V^\top$. Recall that $\langle C_k, \tilde{D}_j \rangle = 0, \forall k, j$ implies that each column of $X$ is orthogonal to

each column of $Z$. Because $r^*, r^{\perp *}, \overline{O}$ are jointly Gaussian, as $\sigma_r \to \infty$, we have

$$\begin{bmatrix} \bar{r} \\ \bar{r}^{\perp} \end{bmatrix} = \operatorname*{argmin}_{(r,r^{\perp})} \frac{1}{2\sigma_w^2} \sum_{n=1}^{N} \left\| y^{(n)} - \sum_{k=1}^{K} r_k x_k^{(n)} - \sum_{j=1}^{J} r_j^{\perp} z_j^{(n)} \right\|^2 + \frac{1}{2\sigma_\epsilon^2} \sum_{j=1}^{J} r_j^{\perp 2}$$

$$= \operatorname*{argmin}_{(r,r^{\perp})} \left\| \begin{bmatrix} \frac{1}{\sigma_w} Y \\ 0 \end{bmatrix} - \begin{bmatrix} \frac{1}{\sigma_w} X & \frac{1}{\sigma_w} Z \\ 0 & \frac{1}{\sigma_\epsilon} I_J \end{bmatrix} \begin{bmatrix} r \\ r^{\perp} \end{bmatrix} \right\|^2 = \begin{bmatrix} (X^\top X)^{-1} X^\top Y \\ (Z^\top Z + \frac{\sigma_w^2}{\sigma_\epsilon^2} I)^{-1} Z^\top Y \end{bmatrix}.$$

Let $a^{(n)} = G_1^{-\frac{1}{2}} G_2 x^{(n)}$, $b^{(n)} = G_1^{-\frac{1}{2}} G_2 z^{(n)}$, $A = \begin{bmatrix} a^{(1)\top} & \cdots & a^{(N)\top} \end{bmatrix}^\top$, $B = \begin{bmatrix} b^{(1)\top} & \cdots & b^{(N)\top} \end{bmatrix}^\top$. We have

$$\begin{aligned} \hat{r} &= \operatorname*{argmin}_{r} \mathbb{E}[\ell(u_r(\tilde{x}), \tilde{y})|\overline{O}] = \operatorname*{argmin}_{r} \frac{1}{N} \sum_{n=1}^{N} \mathbb{E}_{\tilde{y}}[\ell(u_r(\tilde{x}), \tilde{y})|\tilde{x} = x^{(n)}, \overline{O}] \\ &= \operatorname*{argmin}_{r} \sum_{n=1}^{N} u_r(x^{(n)})^\top G_1 u_r(x^{(n)}) + u_r(x^{(n)})^\top G_2 \mathbb{E}[\tilde{y}|\tilde{x} = x^{(n)}, \overline{O}] \\ &= \operatorname*{argmin}_{r} \sum_{n=1}^{N} \frac{1}{4} r^\top a^{(n)\top} a^{(n)} r - \frac{1}{2} r^\top a^{(n)\top} (a^{(n)} \bar{r} + b^{(n)} \bar{r}^{\perp}) \\ &= \bar{r} + A^\dagger B \bar{r}^{\perp} = X^\dagger Y + A^\dagger B (Z^\top Z + \frac{\sigma_w^2}{\sigma_\epsilon^2} I)^{-1} Z^\top Y. \end{aligned} \tag{6}$$

Taking $\sigma_\epsilon \to 0$ and $\sigma_\epsilon \to \infty$ yields

$$\hat{r}_0 = X^\dagger Y, \tag{7}$$
$$\hat{r}_\infty = X^\dagger Y + A^\dagger B Z^\dagger Y. \tag{8}$$

The first part of the theorem then follows because

$$\hat{r}_0 = X^\dagger Y = \operatorname*{argmin}_{r} \|Y - Xr\|^2 = \operatorname*{argmin}_{r} \sum_{n=1}^{N} \left\| y^{(n)} - \sum_{k=1}^{K} r_k x_k^{(n)} \right\|^2.$$

We now prove the second part. Note that

$$\operatorname*{argmin}_{r} \sum_{n=1}^{N} \ell(u_r(x^{(n)}), y^{(n)}) = \operatorname*{argmin}_{r} \sum_{n=1}^{N} u_r(x^{(n)})^\top G_1 u_r(x^{(n)}) + u_r(x^{(n)})^\top G_2 y^{(n)}$$

$$= \operatorname*{argmin}_{r} r^\top A^\top A r - 2r^\top \sum_{n=1}^{N} h^{(n)\top} y^{(n)} = (A^\top A)^{-1} H^\top Y,$$

where $h^{(n)} = G_2^\top G_1^{-1} G_2 x^{(n)}$ and $H = \begin{bmatrix} h^{(1)\top} & \cdots & h^{(N)\top} \end{bmatrix}^\top$. Each $k$th column of $H$

$$h_k = \begin{bmatrix} G_2^\top G_1^{-1} G_2 C_k \phi^{(1)} \\ \vdots \\ G_2^\top G_1^{-1} G_2 C_k \phi^{(N)} \end{bmatrix}$$

is in span$\{\operatorname{col} X, \operatorname{col} Z\}$ because $G_2^\top G_1^{-1} G_2 C_k \in \Re^{M \times Q} = \operatorname{span}\{C_1, \cdots, C_K, \tilde{D}_1, \cdots, \tilde{D}_J\}$. Since the residual $Y' = Y - XX^\dagger Y - ZZ^\dagger Y$ upon projecting $Y$ onto span $\{\operatorname{col} X, \operatorname{col} Z\}$ is orthogonal to the subspace, we have $h_k^\top Y' = 0, \forall k$ and hence $H^\top Y' = 0$. This implies $H^\top Y = H^\top XX^\dagger Y + H^\top ZZ^\dagger Y$. Further, since $a^{(n)\top} a^{(n)} = h^{(n)\top} x^{(n)}$, $a^{(n)\top} b^{(n)} = h^{(n)\top} z^{(n)}, \forall n$, we have

$$\begin{aligned} \hat{r}_\infty &= X^\dagger Y + A^\dagger B Z^\dagger Y = (A^\top A)^{-1} \left( A^\top A X^\dagger Y + A^\top B Z^\dagger Y \right) \\ &= (A^\top A)^{-1} \left( H^\top XX^\dagger Y + H^\top ZZ^\dagger Y \right) = (A^\top A)^{-1} H^\top Y. \end{aligned}$$

$\square$

*Proof of Theorem 2.* Because $\langle \tilde{D}_i, \tilde{D}_j \rangle = 1\{i = j\}$, we have $Z^\top Z = NI$. Plugging this into (6) and comparing the resultant expression with (7) and (8) yield the desired result. $\square$

## References

[1] J.-Y. Audibert. Aggregated estimators and empirical complexity for least square regression. *Annales de l'Institut Henri Poincare Probability and Statistics*, 40(6):685–736, 2004.

[2] P. L. Bartlett and S. Mendelson. Empirical minimization. *Probability Theory and Related Fields*, 135(3):311–334, 2006.

[3] D. Bertsimas and A. Thiele. Robust and data-driven optimization: Modern decision-making under uncertainty. In *Tutorials on Operations Research*. INFORMS, 2006.

[4] O. Besbes, R. Philips, and A. Zeevi. Testing the validity of a demand model: An operations perspective. 2007.

[5] F. Bunea, A. B. Tsybakov, and M. H. Wegkamp. Aggregation for Gaussian regression. *The Annals of Statistics*, 35(4):1674–1697, 2007.

[6] D. Haussler. Decision theoretic generalizations of the PAC model for neural net and other learning applications. *Information and Computation*, 100:78–150, 1992.

[7] K. Kim and N. Timm. *Univariate and Multivariate General Linear Models: Theory and Applications with SAS*. Chapman & Hall/CRC, 2006.

[8] K. E. Muller and P. W. Stewart. *Linear Model Theory: Univariate, Multivariate, and Mixed Models*. Wiley, 2006.

[9] R. S. Sutton and A. G. Barto. *Reinforcement Learning: An Introduction*. MIT Press, Cambridge, MA, 1998.

[10] R. Tibshirani. Regression shrinkage and selection via the lasso. *Journal of Royal Statistical Society*, 1996.

